# Consistent Classification, Firm and Soft

**Yoram Baram***
Department of Computer Science
Technion, Israel Institute of Technology
Haifa 32000, Israel
baram@cs.technion.ac.il

## Abstract

A classifier is called *consistent* with respect to a given set of class-labeled points if it correctly classifies the set. We consider classifiers defined by unions of local separators and propose algorithms for consistent classifier reduction. The expected complexities of the proposed algorithms are derived along with the expected classifier sizes. In particular, the proposed approach yields a consistent reduction of the nearest neighbor classifier, which performs "firm" classification, assigning each new object to a class, regardless of the data structure. The proposed reduction method suggests a notion of "soft" classification, allowing for indecision with respect to objects which are insufficiently or ambiguously supported by the data. The performances of the proposed classifiers in predicting stock behavior are compared to that achieved by the nearest neighbor method.

## 1 Introduction

Certain classification problems, such as recognizing the digits of a hand written zip-code, require the assignment of each object to a class. Others, involving relatively small amounts of data and high risk, call for indecision until more data become available. Examples in such areas as medical diagnosis, stock trading and radar detection are well known. The training data for the classifier in both cases will correspond to firmly labeled members of the competing classes. (A patient may be

either ill or healthy. A stock price may increase, decrease or stay the same). Yet, the classification of new objects need not be firm. (A given patient may be kept in hospital for further observation. A given stock need not be bought or sold every day). We call classification of the first kind "firm" and classification of the second kind "soft". The latter is not the same as training the classifier with a "don't care" option, which would be just another firm labeling option, as "yes" and "no", and would require firm classification. A classifier that correctly classifies the training data is called "consistent". Consistent classifier reductions have been considered in the contexts of the nearest neighbor criterion (Hart, 1968) and decision trees (Holte, 1993, Webb, 1996).

In this paper we present a geometric approach to consistent firm and soft classification. The classifiers are based on unions of local separators, which cover all the labeled points of a given class, and separate them from the others. We propose a consistent reduction of the nearest neighbor classifier and derive its expected design complexity and the expected classifier size. The nearest neighbor classifier and its consistent derivatives perform "firm" classification. Soft classification is performed by unions of maximal–volume spherical local separators. A domain of indecision is created near the boundary between the two sets of class–labeled points, and in regions where there is no data. We propose an economically motivated *benefit* function for a classifier as the difference between the probabilities of success and failure. Employing the respective benefit functions, the advantage of soft classification over firm classification is shown to depend on the rate of indecision. The performances of the proposed algorithms in predicting stock behavior are compared to those of the nearest neighbor method.

## 2 Consistent Firm Classification

Consider a finite set of points $X = \{x^{(i)}, i = 1, \ldots, N\}$ in some subset of $R^n$, the real space of dimension $n$. Suppose that each point of $X$ is assigned to one of two classes, and let the corresponding subsets of $X$, having $N_1$ and $N_2$ points, respectively, be denoted $X_1$ and $X_2$. We shall say that the two sets are labeled $L_1$ and $L_2$, respectively. It is desired to divide $R^n$ into labeled regions, so that new, unlabeled points can be assigned to one of the two classes.

We define a *local separator* of a point $x$ of $X_1$ with respect to $X_2$ as a convex set, $s(x|2)$, which contains $x$ and no point of $X_2$. A *separator family* is defined as a rule that produces local separators for class–labeled points.

We call the set of those points of $R^n$ that are closer to a point $x \in X_1$ than to any point of $X_2$ the *minimum–distance* local separator of $x$ with respect to $X_2$.

We define the *local clustering degree*, $c$, of the data as the expected fraction of data points that are covered by a local minimum–distance separator.

The *nearest neighbor* criterion extends the class assignment of a point $x \in X_1$ to its minimum–distance local separator. It is clearly a consistent and firm classifier whose memory size is $O(N)$.

Hart's **Condensed Nearest Neighbor (CNN)** classifier (Hart, 1968) is a *consistent subset* of the data points that correctly classifies the entire data by the nearest neighbor method. It is not difficult to show that the complexity of the algorithm

proposed by Hart for finding such a subset is $O(N^3)$. The expected memory requirement (or classifier size) has remained an open question.

We propose the following **Reduced Nearest Neighbor (RNN)** classifier: include a labeled point in the consistent subset only if it is not covered by the minimum-distance local separator of any of the points of the same class already in the subset.

It can be shown (Baram, 1996) that the complexity of the RNN algorithm is $O(N^2)$. and that the expected classifier size is $O(\log_{1/(1-c)} N)$. It can also be shown that the latter bounds the expected size of the CNN classifier as well.

It has been suggested that the utility of the Occam's razor in classification would be (Webb, 1996):

"Given a choice between two plausible classifiers that perform identically on the data set, the simpler classifier is expected to classify correctly more objects outside the training set".

The above statement is disproved by the CNN and the RNN classifiers, which are strict consistent reductions of the nearest neighbor classifier, likely to produce more errors.

## 3   Soft Classification: Indecision Pays, Sometimes

When a new, unlabeled, point is closely surrounded by many points of the same class, its assignment to the same class can be said to be unambiguously supported by the data. When a new point is surrounded by points of different classes, or when it is relatively far from any of the labeled points, its assignment to either class can be said to be unsupported or ambiguously supported by the data. In the latter cases, it may be more desirable to have a certain indecision domain, where new points will not be assigned to a class. This will translate into the creation of indecision domains near the boundary between the two sets of labeled points and where there is no data.

We define a *separator* $S(1|2)$ of $X_1$ with respect to $X_2$ as a set that includes $X_1$ and excludes $X_2$.

Given a separator family, the union of local separators $s(x^{(i)}|2)$ of the points $x^{(i)}$, $i = 1 \ldots, N_1$, of $X_1$ with respect to $X_2$,

$$S(1|2) = \cup_{x^{(i)} \in X_1} s(x^{(i)}|2) \tag{1}$$

is a separator of $X_1$ with respect to $X_2$. It consists of $N_1$ local separators.

Let $X_{1,c}$ be a subset of $X_1$. The set

$$S_c(1|2) = \cup_{x^{(i)} \in X_{1,c}} s(x^{(i)}|2) \tag{2}$$

will be called a *consistent* separator of $X_1$ with respect to $X_2$ if it contains all the points of $X_1$. The set $X_{1,c}$ will then be called a *consistent subset* with respect to the given separator family.

Let us extend the class assignment of each of the labeled points to a local separator of a given family and maximize the volume of each of the local separators without

including in it any point of the competing class. Let $S_c(1|2)$ and $S_c(2|1)$ be consistent separators of the two sets, consisting of maximal–volume (or, simply, *maximal*) local separators of labeled points of the corresponding classes. The intersection of $S_c(1|2)$ and $S_c(2|1)$ defines a conflict and will be called a *domain of ambiguity of the first kind*. A region uncovered by either separator will be called a *domain of ambiguity of the second kind*. The union of the domains of ambiguity will be designated the *domain of indecision*. The remainders of the two separators, excluding their intersection, define the conflict–free domains assigned to the two classes.

The resulting "soft" classifier rules out hard conflicts, where labeled points of one class are included in the separator of the other. Yet, it allows for indecision in areas which are either claimed by both separators or claimed by neither.

Let the true class be denoted $y$ (with possible values, e.g., y=1 or y=2) and let the classification outcome be denoted $\hat{y}$. Let the probabilities of decision and indecision by the soft classifier be denoted $P_d$ and $P_{id}$, respectively (of course, $P_{id} = 1 - P_d$), and let the probabilities of correct and incorrect decisions by the firm and the soft classifiers be denoted $P_{\text{firm}}\{\hat{y} = y\}$, $P_{\text{firm}}\{\hat{y} \neq y\}$, $P_{\text{soft}}\{\hat{y} = y\}$ and $P_{\text{soft}}\{\hat{y} \neq y\}$, respectively. Finally, let the joint probabilities of a decision being made by the soft classifier and the correctness or incorrectness of the decision be denoted, respectively, $P_{\text{soft}}\{d, \hat{y} = y\}$ and $P_{\text{soft}}\{d, \hat{y} \neq y\}$ and let the corresponding conditional probabilities be denoted $P_{\text{soft}}\{\hat{y} = y \mid d\}$ and $P_{\text{soft}}\{\hat{y} \neq y \mid d\}$, respectively.

We define the *benefit* of using the firm classifier as the difference between the probability that a point is classified correctly by the classifier and the probability that it is misclassified:

$$B_{\text{firm}} = P_{\text{firm}}\{\hat{y} = y\} - P_{\text{firm}}\{\hat{y} \neq y\} = 2P_{\text{firm}}\{\hat{y} = y\} - 1. \qquad (3)$$

This definition is motivated by economic consideration: the profit produced by an investment will be, on average, proportional to the benefit function. This will become more evident in a later section, were we consider the problem of stock trading.

For a soft classifier, we similarly define the benefit as the difference between the probability of a correct classification and that of an incorrect one (which, in an economic context, assumes that indecision has no cost, other than the possible loss of profit). Now, however, these probabilities are for the joint events that a classification is made, and that the outcome is correct or incorrect, respectively:

$$\begin{aligned} B_{\text{soft}} &= P_{\text{soft}}\{d, \hat{y} = y\} - P_{\text{soft}}\{d, \hat{y} \neq y\} \\ &= [2P_{\text{soft}}\{\hat{y} = y \mid d\} - 1]P_d \end{aligned} \qquad (4)$$

Soft classification will be more beneficial than firm classification if $B_{\text{soft}} > B_{\text{firm}}$, which may be written as

$$P_{id} < 1 - \frac{P_{\text{firm}}\{\hat{y} = y\} - 0.5}{P_{\text{soft}}\{\hat{y} = y \mid d\} - 0.5} \qquad (5)$$

For the latter to be a useful condition, it is necessary that $P_{\text{firm}}\{\hat{y} = y\} > 0.5$, $P_{\text{soft}}\{\hat{y} = y \mid d\} > 0.5$ and $P_{\text{soft}}\{\hat{y} = y \mid d\} > P_{\text{firm}}\{\hat{y} = y\}$. The latter will be normally satisfied, since points of the same class can be expected to be denser under the corresponding separator than in the indecision domain. In other words,

the error ratio produced by the soft classifier on the decided cases can be expected to be smaller than the error ratio produced by the firm classifier, which decides on all the cases. The satisfaction of condition (5) would depend on the geometry of the data. It will be satisfied for certain cases, and will not be satisfied for others. This will be numerically demonstrated for the stock trading problem.

The maximal local spherical separator of $x$ is defined by the open sphere centered at $x$, whose radius $r(x|2)$ is the distance between $x$ and the point of $X_2$ nearest to $x$. Denoting by $s(x, r)$ the sphere of radius $r$ in $R^n$ centered at $x$, the maximal local separator is then $s_M(x|2) = s(x, r(x|2))$.

A separator construction algorithm employing maximal local spherical separators is described below. Its complexity is clearly $O(N^2)$.

Let $\tilde{X}_1 = X_1$. For each of the points $x^{(i)}$ of $\tilde{X}_1$, find the minimal distance to the points of $X_2$. Call it $r(x^{(i)}|2)$. Select the point $x^{(i)}$ for which $r(x^{(i)}|2) \geq r(x^{(j)}|2)$, $j \neq i$, for the consistent subset. Eliminate from $\tilde{X}_1$ all the points that are covered by $s_M(x^{(i)}|2)$. Denote the remaining set $\tilde{X}_1$. Repeat the procedure while $\tilde{X}_1$ is non–empty. The union of the maximal local spherical separators is a separator for $X_1$ with respect to $X_2$.

## 4    Example: Firm and soft prediction of stock behaviour

Given a sequence of $k$ daily trading ("close") values of a stock, it is desired to predict whether the next day will show an increase or a decrease with respect to the last day in the sequence. Records for ten different stocks, each containing, on average, 1260 daily values were used. About 60 percent of the data were used for training and the rest for testing. The CNN algorithm reduced the data by 40% while the RNN algorithm reduced the data by 35%. Results are show in Fig. 1. It can be seen that, on average, the nearest neighbor method has produced the best results. The performances of the CNN and the RNN classifiers (the latter producing only slightly better results) are somewhat lower. It has been argued that performance within a couple of percentage points by a reduced classifier supports the utility of Occam's razor (Holte, 1993). However, a couple of percentage points can be quite meaningful in stock trading.

In order to evaluate the utility of soft classification in stock trading, let the prediction success rate of a firm classifier, be denoted $f$ and that of a soft classifier for the decided cases $s$. For a given trade, let the gain or loss per unit invested be denoted $q$, and the rate of indecision of the soft classifier $ir$. Suppose that, employing the firm classifier, a stock is traded once every day (say, at the "close" value), and that, employing the soft classifier, it is traded on a given day only if a trade is decided by the classifier (that is, the input does not fall in the indecision domain). The expected profit for $M$ days per unit invested is $2(f - 0.5)qM$ for the firm classifier and $2(s - 0.5)q(1 - ir)M$ for the soft classifier (these values disregard possible commission and slippage costs). The soft classifier will be preferred over the firm one if the latter quantity is greater than the former, that is, if

$$ir < 1 - \frac{f - 0.5}{s - 0.5} \tag{6}$$

which is the sample representation of condition (5) for the stock trading problem.

| | NN | CNN | RNN | soft classifier | | |
| | | | | indecision | success | benefit |
|---|---|---|---|---|---|---|
| xaip | 61.4% | 58.4% | 57.0% | 48.4% | 70.1% | - |
| xaldnf | 51.7% | 52.3% | 50.1% | 74.6% | 51.3% | - |
| xddddf | 52.0% | 49.6% | 51.8% | 44.3% | 53.3% | - |
| xdssi | 48.3% | 47.7% | 48.6% | 43.6% | 52.6% | + |
| xecilf | 53.0% | 50.9% | 52.6% | 47.6% | 48.8% | - |
| xedusf | 80.7% | 74.7% | 76.3% | 30.6% | 89.9% | - |
| xelbtf | 53.7% | 55.6% | 52.5% | 42.2% | 50.1% | - |
| xetz | 66.0% | 61.0% | 61.0% | 43.8% | 68.6% | - |
| xelrnf | 51.5% | 49.0% | 49.2% | 39.2% | 56.0% | + |
| xelt | 85.6% | 82.7% | 84.2% | 32.9% | 93.0% | - |
| Average | 60.4% | 58.2% | 58.3% | 44.7% | 63.4% | |

Figure 1: Success rates in the prediction of rize and fall in stock values.

Results for the soft classifier, applied to the stock data, are presented in Fig. 1. The indecision rates and the success rates in the decided cases are then specified along with a benefit sign. A positive benefit represents a satisfaction of condition (6), with $ir$, $f$ and $s$ replaced by the corresponding sample values given in the table. This indicates a higher profit in applying the soft classifier over the application of the nearest neighbor classifier. A negative benefit indicates that a higher profit is produced by the nearest neighbor classifier. It can be seen that for two of the stocks (xdssi and xelrnf) soft classification has produced better results than firm classification, and for the remaining eight stocks firm classification by the nearest neighbor method has produced better results.

## 5  Conclusion

Solutions to the consistent classification problem have been specified in terms of local separators of data points of one class with respect to the other. The expected complexities of the proposed algorithms have been specified, along with the expected sizes of the resulting classifiers. Reduced consistent versions of the nearest neighbor classifier have been specified and their expected complexities have been derived. A notion of "soft" classification has been introduced an algorithm for its implementation have been presented and analyzed. A criterion for the utility of such classification has been presented and its application in stock trading has been demonstrated.

### Acknowledgment

The author thanks Dr. Amir Atiya of Cairo University for providing the stock data used in the examples and for valuable discussions of the corresponding results.

## Footnotes

*Presently a Senior Research Associate of the National Research Council at M. S. 210-9, NASA Ames Research Center, Moffett Field, CA 94035, on sabbatical leave from the Technion.

## References

Baram Y. (1996) Consistent Classification, Firm and Soft, CIS Report No. 9627, Center for Intelligent Systems, Technion, Israel Institute of Technology, Haifa 32000, Israel.

Baum, E. B. (1988) On the Capabilities of Multilayer Perceptrons, J. Complexity, Vol. 4, pp. 193 – 215.

Hart, P. E. (1968) The Condensed Nearest Neighbor Rule, IEEE Trans. on Information Theory, Vol. IT–14, No. 3, pp. 515 – 516.

Holte, R. C. (1993) Very Simple Classification Rules Perform Well on Most Commonly Used databases, Machine Learning, Vol. 11, No. 1 pp. 63 – 90.

Rosenblatt, F. (1958) The Perceptron: A Probabilistic Model for Information Storage and Organization in the Brain, Psychological Review, Vol. 65, pp. 386 – 408.

Webb, G. I. (1996) Further Experimental Evidence against the Utility of Occam's Razor, J. of Artificial Intelligence Research 4, pp. 397 – 147.
